# The Information-Form Data Association Filter

**Brad Schumitsch, Sebastian Thrun, Gary Bradski, and Kunle Olukotun**

Stanford AI Lab
Stanford University, Stanford, CA 94305

## Abstract

This paper presents a new filter for online data association problems in high-dimensional spaces. The key innovation is a representation of the data association posterior in information form, in which the "proximity" of objects and tracks are expressed by numerical links. Updating these links requires linear time, compared to exponential time required for computing the exact posterior probabilities. The paper derives the algorithm formally and provides comparative results using data obtained by a real-world camera array and by a large-scale sensor network simulation.

## 1 Introduction

This paper addresses the problem of data association in online object tracking [6]. The data association problem arises in a large number of application domains, including computer vision, robotics, and sensor networks.

Our setup assumes an online tracking system that receives two types of data: *sensor data*, conveying information about the identity or type of objects that are being tracked; and *transition data*, characterizing the uncertainty introduced through the tracker's inability to reliably track individual objects over time. The setup is motivated by a camera network which we recently deployed in our lab. Here sensor data relates to the color of clothing of individual people, which enables us to identify them. Tracks are lost when people walk too closely together, or when they occlude each other.

We show that the standard probabilistic solution to the discrete data association problem requires exponential update time and exponential memory. This is because each data association hypothesis is expressed by a permutation matrix that assigns computer-internal tracks to objects in the physical world. An optimal filter would therefore need to maintain a probability distribution over the space of all permutation matrices, which grows exponentially with $N$, the number of objects in the world. The common remedy involves the selection of a small number $K$ of likely hypotheses. This is the core of numerous widely-used multi-hypothesis tracking algorithms [9, 1]. More recent solutions involve particle filters [3], which maintain stochastic samples of hypotheses. Both of these techniques are very effective for small N, but the number of hypothesis they require grows exponentially with $N$.

This paper provides a filter algorithm that scales to much larger problems. This filter maintains an information matrix $\Omega$ of size $N \times N$, which relates tracks to physical objects in the world. The rows of $\Omega$ correspond to object identities, the columns to the tracks of the tracker. $\Omega$ is a matrix in *information form*, that is, it can be thought of as a non-normalized log-probability.

Fig. 1a shows an example. The highlighted first column corresponds to track 1 in the tracker. The numerical values in this column suggest that this track is most strongly

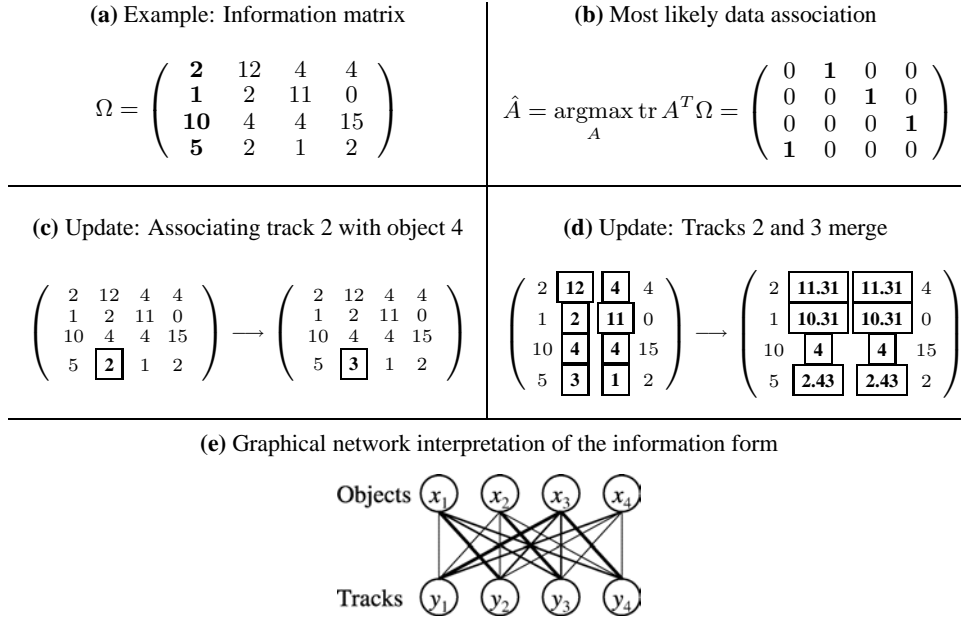

**(a)** Example: Information matrix

$$\Omega = \begin{pmatrix} \mathbf{2} & 12 & 4 & 4 \\ \mathbf{1} & 2 & 11 & 0 \\ \mathbf{10} & 4 & 4 & 15 \\ \mathbf{5} & 2 & 1 & 2 \end{pmatrix}$$

**(b)** Most likely data association

$$\hat{A} = \operatorname*{argmax}_{A} \operatorname{tr} A^T \Omega = \begin{pmatrix} 0 & \mathbf{1} & 0 & 0 \\ 0 & 0 & \mathbf{1} & 0 \\ 0 & 0 & 0 & \mathbf{1} \\ \mathbf{1} & 0 & 0 & 0 \end{pmatrix}$$

**(c)** Update: Associating track 2 with object 4

$$\begin{pmatrix} 2 & 12 & 4 & 4 \\ 1 & 2 & 11 & 0 \\ 10 & 4 & 4 & 15 \\ 5 & \boxed{2} & 1 & 2 \end{pmatrix} \longrightarrow \begin{pmatrix} 2 & 12 & 4 & 4 \\ 1 & 2 & 11 & 0 \\ 10 & 4 & 4 & 15 \\ 5 & \boxed{3} & 1 & 2 \end{pmatrix}$$

**(d)** Update: Tracks 2 and 3 merge

$$\begin{pmatrix} 2 & \boxed{12} & \boxed{4} & 4 \\ 1 & \boxed{2} & \boxed{11} & 0 \\ 10 & \boxed{4} & \boxed{4} & 15 \\ 5 & \boxed{3} & \boxed{1} & 2 \end{pmatrix} \longrightarrow \begin{pmatrix} 2 & \boxed{11.31} & \boxed{11.31} & 4 \\ 1 & \boxed{10.31} & \boxed{10.31} & 0 \\ 10 & \boxed{4} & \boxed{4} & 15 \\ 5 & \boxed{2.43} & \boxed{2.43} & 2 \end{pmatrix}$$

**(e)** Graphical network interpretation of the information form

**Figure 1**: Illustration of the information form filter for data association in object tracking

associated with object 3, since the value 10 dominates all other values in this column. Thus, looking at column 1 of $\Omega$ in isolation would have us conclude that the most likely association of track 1 is object 3. However, the most likely permutation matrix is shown in Fig. 1b; from all possible data association assignments, this matrix receives the highest score. Its score is $\operatorname{tr} \hat{A}^T \Omega = 5 + 12 + 11 + 15 = 43$ (here "tr" denotes the trace of a matrix). This permutation matrix associates object 3 with track 4, while associating track 1 with object 4.

The key question now pertains to the construction of $\Omega$. As we shall see, the update operations for $\Omega$ are simple and parallelizable. Suppose we receive a measurement that associates track 2 with object 4 (e.g., track 2's hair color appears to be the same as person 4's hair color in our camera array). As a result, our approach adds a value to the element in $\Omega$ that links object 4 and track 2, as illustrated in Fig. 1c (the exact magnitude of this value will be discussed below). Similarly, suppose our tracker is unable to distinguish between objects 2 and 3, perhaps because these objects are so close together in a camera image that they cannot be tracked individually. Such a situation leads to a new information matrix, in which both columns assume the same values, as illustrated in Fig. 1d. The exact values in this new information matrix are the result of an exponentiated averaging explained below. All of these updates are easily parallelized, and hence are applicable to a decentralized network of cameras. The exact update and inference rules are based on a probabilistic model that is also discussed below.

Given the importance of data association, it comes as no surprise that our algorithm is related to a rich body of prior work. The data association problem has been studied as an *offline* problem, in which all data is memorized and inference takes place after data collection. There exists a wealth of powerful methods, such as RANSAC [4] and MCMC [6, 2], but those are inherently offline and their memory requirements increase over time. The dominant online, or filter, paradigm involves the selection of $K$ representative samples of the data association matrix, but such algorithms tend to work only for small $N$ [11]. Relatively little work has focused on the development of compact sufficient statistics for data association. One alternative $O(N^2)$ technique to the one proposed here was explored in [8]. This technique uses doubly stochastic matrices, which are computationally hard to maintain. The first mention of information filters is in [8], but the update rules there were

computationally less efficient (in $O(N^4)$) and required central optimization.

The work in this paper does not address the continuous-valued aspects of object tracking. Those are very well understood, and information representations have been successfully applied [5, 10].

Information representations are popular in the field of graphical networks. Our approach can be viewed as a learning algorithm for a Markov network [7] of a special topology, where any track and any object are connected by an edge. Such a network is shown in Fig. 1e. The filter update equations manipulate the strength of the edges based on data.

## 2   Problem Setup and Bayes Filter Solution

We begin with a formal definition of the data association problem and derive the obvious but inefficient Bayes filter solution. Throughout this paper, we make the closed world assumption, that is, there are always the same $N$ known objects in the world.

### 2.1   Data Association

We assume that we are given a tracking algorithm that maintains $N$ internal tracks of the moving objects. Due to insufficient information, this assumed tracking algorithm does not always know the exact mapping of identities to internal tracks. Hence, the same internal track may correspond to different identities at different times.

The data association problem is the problem of assigning these $N$ tracks to $N$ objects. Each data association hypothesis is characterized by a permutation matrix of the type shown in Fig. 1b. The columns of this matrix correspond to the internal tracks, and the rows to the objects. We will denote the data association matrix by $A$ (not to be confused with the information matrix $\Omega$). In our closed world, $A$ is always a permutation matrix; hence all elements are 0 or 1. There are exponentially many permutation matrices, which is a reason why data association is considered a hard problem.

### 2.2   Identity Measurement

The correct data association matrix $A$ is unobservable. Instead, the sensors produce local information about the relation of individual tracks to individual objects. We will denote sensor measurements by $z_j$, where $j$ is the index of the corresponding track. Each $z_j = \{z_{ij}\}$ specifies a local probability distribution in the corresponding object space:

$$p(x_i = y_j \mid z_j) \quad = \quad z_{ij} \qquad \text{with} \quad \sum_i z_{ij} = 1 \tag{1}$$

Here $x_i$ is the $i$-th object in the world, and $y_j$ is the $j$-th track.

The measurement in our introductory example (see Fig. 1c) was of a special form, in that it elevated one specific correspondence over the others. This occurs when $z_{ij} = \alpha$ for some $\alpha \approx 1$, and $z_{kj} = \frac{1-\alpha}{N-1}$ for all $k \neq i$. Such a measurement arises when the tracker receives evidence that a specific track $y_j$ corresponds with high likelihood to a specific object $x_i$. Specifically, the measurement likelihood of this correspondence is $\alpha$, and the error probability is $1 - \alpha$.

### 2.3   State Transitions

As time passes by, our tracker may confuse tracks, which is a loss of information with respect to the data association. The tracker confusing two objects amounts to a random flip of two columns in the data association matrix $A$.

The model adopted in this paper generalizes this example to arbitrary distributions over permutations of the columns in $A$. Let $\{B_1, \ldots, B_M\}$ be a set of permutation matrices, and $\{\beta_1, \ldots, \beta_M\}$ with $\sum_m \beta_m = 1$ be a set of associated probabilities. The "true" permutation matrix undergoes a random transition from $A$ to $A\,B_m$ with probability $\beta_m$:

$$A \quad \xrightarrow{\text{prob}=\beta_m} \quad A\,B_m \tag{2}$$

The sets $\{B_1, \ldots, B_M\}$ and $\{\beta_1, \ldots, \beta_M\}$ are given to us by the tracker. For the example in Fig. 1d, in which tracks 2 and 3 merge, the following two permutation matrices will implement such a merge:

$$B_1 \;=\; \begin{pmatrix} 1 & 0 & 0 & 0 \\ 0 & 1 & 0 & 0 \\ 0 & 0 & 1 & 0 \\ 0 & 0 & 0 & 1 \end{pmatrix}; \beta_1 = 0.5 \qquad B_2 \;=\; \begin{pmatrix} 1 & 0 & 0 & 0 \\ 0 & 0 & 1 & 0 \\ 0 & 1 & 0 & 0 \\ 0 & 0 & 0 & 1 \end{pmatrix}; \beta_2 = 0.5 \quad (3)$$

The first such matrix leaves the association unchanged, whereas the second swaps columns 2 and 3. Since $\beta_1 = \beta_2 = 0.5$, such a swap happens exactly with probability 0.5.

### 2.4 Inefficient Bayesian Solution

For small $N$, the data association problem now has an obvious Bayes filter solution. Specifically, let $\mathcal{A}$ be the space of all permutation matrices. The Bayesian filter solves the identity tracking problem by maintaining a probabilistic belief over the space of all permutation matrices $A \in \mathcal{A}$. For each $A$, it maintains a posterior probability denoted $p(A)$. This probability is updated in two different ways, reminiscent of the measurement and state transition updates in DBNs and EKFs.

The measurement step updates the belief in response to a measurement $z_j$. This update is an application of Bayes rule:

$$p(A) \quad\longleftarrow\quad \frac{1}{L}\, p(A) \sum_i a_{ij}\, z_{ij} \tag{4}$$

$$\text{with} \quad L \quad = \quad \sum_{\bar{A}} p(\bar{A}) \sum_i \bar{a}_{ij}\, z_{ij} \tag{5}$$

Here $a_{ij}$ denotes the $ij$-th element of the matrix $A$. Because $A$ is a permutation matrix, only one element in the sum over $i$ is non-zero (hence there is not really a summation here).

The state transition updates the belief in accordance with the permutation matrices $B_m$ and associated probabilities $\beta_m$ (see Eq. 2):

$$p(A) \quad\longleftarrow\quad \sum_m \beta_m\, p(A\, B_m^T) \tag{6}$$

We use here that the inverse of a permutation matrix is its transpose.

This Bayesian filter is an exact solution to our identity tracking problem. Its problem is complexity: there are $N!$ permutation matrices $A$, and we have to compute probabilities for all of them. Thus, the exact filter is only applicable to problems with small $N$. Even if we want to keep track of $K \ll N$ *likely* permutations—as attempted by filters like the multi-hypothesis EKF or the particle filter—the required number of tracks $K$ will generally have to scale exponentially with $N$ (albeit at a slower rate). This exponential scaling renders the Bayesian filter ultimately inapplicable to the identity tracking problem with large $N$.

## 3 The Information-Form Solution

Our data association filter represents the posterior in condensed form, using an $N \times N$ information matrix. As a result, it requires linear update time and quadratic memory, instead of the exponential time and memory requirements of the Bayes filter.

However, we give two caveats regarding our method: it is approximate, and it does not maintain probabilities. The approximation is the result of a Jensen approximation, which we will show is empirically accurate. The calculation of probabilities from an information matrix requires inference, and we will provide several options for performing this inference.

### 3.1 The Information Matrix

The information matrix, denoted $\Omega$, is a matrix of size $N \times N$ whose elements are non-negative. $\Omega$ induces a probability distribution over the space of all data association matrices

$\mathcal{A}$, through the following definition:

$$p(A) \quad = \quad \frac{1}{Z} \ \exp \operatorname{tr} A \, \Omega \quad \text{with} \quad Z \ = \ \sum_{A} \exp \operatorname{tr} A \, \Omega \qquad (7)$$

Here $\operatorname{tr}$ is the trace of a matrix, and $Z$ is the partition function.

Computing the posterior probability $p(A)$ from $\Omega$ is hard, due to the difficulty of computing the partition function $Z$. However, as we shall see, maintaining $\Omega$ is surprisingly easy, and it is also computationally efficient.

## 3.2 Measurement Update in Information Form

In information form, the measurement update is a local addition of the form:

$$\Omega \quad \longleftarrow \quad \Omega + \begin{pmatrix} 0\cdots0 & \log z_{1j} & 0\cdots0 \\ \vdots\ddots\vdots & \vdots & \vdots\ddots\vdots \\ 0\cdots0 & \log z_{1N} & 0\cdots0 \end{pmatrix} \qquad (8)$$

This follows directly from Eq. 4. The complexity of this update is $O(N)$.

Of particular interest is the case where one specific association was affirmed with probability $z_{ij} = \alpha$, while all others were true with the error probability $z_{kj} = \frac{1-\alpha}{N-1}$. Then the update is of the form

$$\Omega \quad \longleftarrow \quad \Omega + \begin{pmatrix} 0\cdots0 & c & 0\cdots0 \\ \vdots\ddots\vdots & \vdots & \vdots\ddots\vdots \\ 0\cdots0 & c & 0\cdots0 \\ \vdots\ddots\vdots & \log\alpha & \vdots\ddots\vdots \\ 0\cdots0 & c & 0\cdots0 \\ \vdots\ddots\vdots & \vdots & \vdots\ddots\vdots \\ 0\cdots0 & c & 0\cdots0 \end{pmatrix} \quad \text{with} \quad c = \log \frac{1-\alpha}{N-1} \qquad (9)$$

However, since $\Omega$ is a non-normalized matrix (it is normalized via the partition function $Z$ in Eq. 7), we can modify $\Omega$ as long as $\exp \operatorname{tr} A \, \Omega$ is changed by the same factor for any $A$. In particular, we can subtract $c$ from an entire column in $\Omega$; this will affect the result of $\exp \operatorname{tr} A \, \Omega$ by a factor of $\exp c$, which is independent of $A$ and hence will be subsumed by the normalizer $Z$. This allows us to perform a more efficient update

$$\omega_{ij} \quad \longleftarrow \quad \omega_{ij} + \log\alpha - \log \frac{1-\alpha}{N-1} \qquad (10)$$

where $\omega_{ij}$ is the $ij$-th element of $\Omega$. This update is indeed of the form shown in Fig. 1c. It requires $O(1)$ time, is entirely local, and is an exact realization of Bayes rule in information form.

## 3.3 State Transition Update in Information Form

The state transition update is also simple, but it is approximate. We show that using a Jensen bound, we obtain the following update for the information matrix:

$$\Omega \quad \longleftarrow \quad \log \sum_{m} \beta_m \, B_m^T \, \exp \Omega \qquad (11)$$

Here the expression "$\exp\Omega$" denotes a component-wise exponentiation of the matrix $\Omega$; the result is also a matrix. This update implements a "dual" of a geometric mean; here the exponentiation is applied to the individual elements of this mean, and the logarithm is applied to the result. It is important to notice that this update only affects elements in $\Omega$ that might be affected by a permutation $B_m$; all others remain the same.

A numerical example of this update was given in Fig. 1d, assuming the permutation matrices in Eq. 3. The values there are the result of applying this update formula. For example, for the first row we get $\log \frac{1}{2}(\exp 12 + \exp 4) = 11.3072$.

The derivation of this update formula is straightforward. We begin with Eq. 6, written in logarithmic form. The transformations rely heavily on the fact that $A$ and $B_m$ are permutation matrices. We use the symbol "$\mathrm{tr}^*$" for a multiplicative version of the matrix trace, in which all elements on the diagonal are multiplied.

$$
\begin{aligned}
\log p(A) \quad &\longleftarrow \quad \log \sum_m \beta_m \, p(A \, B_m^T) \\
&= \quad \text{const.} + \log \sum_m \beta_m \, \exp \mathrm{tr}\, A \, B_m^T \, \Omega \\
&= \quad \text{const.} + \log \sum_m \beta_m \, \mathrm{tr}^* \exp A \, B_m^T \, \Omega \\
&= \quad \text{const.} + \log \sum_m \beta_m \, \mathrm{tr}^* A \, B_m^T \, \exp \Omega \\
&\leq \quad \text{const.} + \log \mathrm{tr}^* A \, \sum_m \beta_m \, B_m^T \, \exp \Omega \\
&= \quad \text{const.} + \mathrm{tr}\, A \left[ \log \sum_m \beta_m \, B_m^T \, \exp \Omega \right] \qquad (12)
\end{aligned}
$$

The result is of the form of (the logarithm of) Eq. 7. The expression in brackets is equivalent to the right-hand side of the update Eq. 11. A benefit of this update rule is that it only affects columns in $\Omega$ that are affected by a permutation $B_m$; all other columns are unchanged.

We note that the approximation in this derivation is the result of applying a Jensen bound. As a result, we gain a compact closed-form solution to the update problem, but the state transition step may sacrifice information in doing so (as indicated by the "$\leq$" sign). In our experimental results section, however, we find that this approximation is extremely accurate in practice.

## 4  Computing the Data Association

The previous section formally derived our update rules, which are simple and local. We now address the problem of recovering actual data association hypotheses from the information matrix, along with the associated probabilities.

We consider three cases: the computation of the most likely data association matrix as illustrated in Fig. 1b; the computation of a relative probability of the form $p(A)/p(A')$; and the computation of an absolute probability or expectation.

To recover $\mathrm{argmax}_A \, p(A)$, we need only solve a linear program.

Relative probabilities are also easy to recover. Consider, for example, the quotient of the probability $p(A)/p(A')$ for two identity matrices $A$ and $A'$. When calculating this quotient from Eq. 7, the normalizer $Z$ cancels out:

$$
\frac{p(A)}{p(A')} \quad = \quad \exp \mathrm{tr}(A - A') \, \Omega \qquad (13)
$$

Absolute probabilities and expectations are generally the most difficult to compute. This is because of the partition function $Z$ in Eq. 7, whose exact calculation requires considering $N!$ permutation matrices.

Our approximate method for recovering probabilities/expectations is based on the Metropolis algorithm. Specifically, consider the expectation of a function $f$:

$$
E[f(A)] \quad = \quad \sum_A f(A) \, p(A) \qquad (14)
$$

Our method approximates this expression through a finite sample of matrices $A^{[1]}, A^{[2]}, \ldots$, using Metropolis and the proposal distribution defined in Eq. 13. This proposal generates excellent results for simple functions $f$ (e.g., the marginal of a single identity). For more

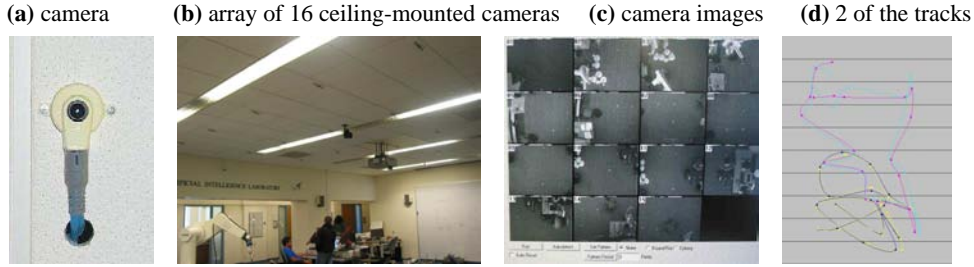

**(a)** camera    **(b)** array of 16 ceiling-mounted cameras    **(c)** camera images    **(d)** 2 of the tracks

**Figure 2**: The camera array, part of the common area in the Stanford AI Lab. Panel (d) compares our esitmate with ground truth for two of the tracks. The data association is essentially correct at all times.

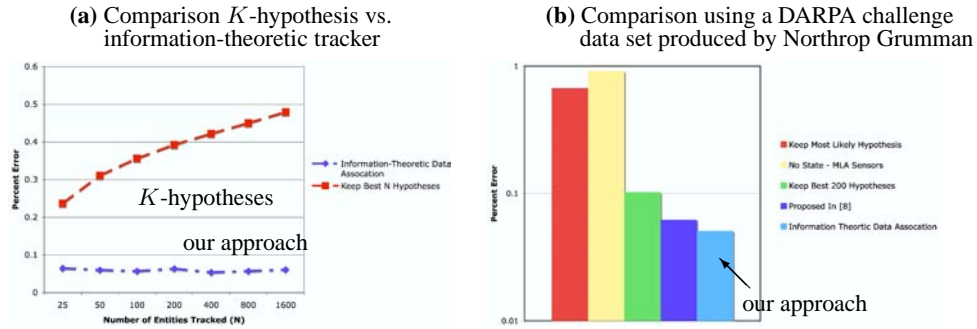

**(a)** Comparison $K$-hypothesis vs. information-theoretic tracker        **(b)** Comparison using a DARPA challenge data set produced by Northrop Grumman

**Figure 3**: Results for our approach information-form filter the common multi-hypothesis approach for (a) synthetic data and (b) a DARPA challenge data set. The comparison (b) involves additional algorithms, including one published in [8].

complex functions $f$, we refer the reader to improved proposal distributions that have been found to be highly efficient in related problems [6, 2].

## 5    Experimental Results

To evaluate this algorithm, we deployed a network of ceiling-mounted cameras in our lab, shown in Fig. 2. We used 16 cameras to track individuals walking through the lab. The tracker uses background subtraction to find blobs and uses a color histogram to classify these blobs. Only when two or more people come very close to each other might the tracker lose track of individual people. We find that for $N = 5$ our method tracks people nearly perfectly, but so does the full-blown Bayesian solution, as well as the $K$-best multi-hypothesis method that is popular in the tracking literature.

To investigate scaling to larger $N$, we compared our approach on two data sets: a synthetic one with up to $N = 1,600$ objects, and a dataset using an sensor network simulation provided to us by Northrop Grumman through an ongoing DARPA program. The latter set is thought to be realistic. It was chosen because it involves a large number ($N = 200$) of moving objects, whose motion patterns come from a behavioral model. In all cases, we measured the number of objects mislabeled in the maximum likelihood hypothesis (as found by solving the LP). All results are averaged over 50 runs.

The comparison in Fig. 3a shows that our approach outperforms the traditional $K$-best hypothesis approach (with $K = N$) by a large margin. Furthermore, our approach seems to be unaffected by $N$, the number of entities in the environment, whereas the traditional approach deteriorates. This comes as no surprise, since the traditional approach requires increasing numbers of samples to cover the space of all data associations. The results in Fig. 3b compare (from left to right), the most likely hypothesis, the most recent sensor measurement, the $K$-best approach with $K = 200$, an approach proposed in [8], and our approach. Notice that this plot is in log-form.

No comparisons were attempted with offline techniques, such as the ones in [4, 6], because the data sets used here are quite large and our interest is online filtering.

# 6 Conclusion

We have provided an information form algorithm for the data association problem in object tracking. The key idea of this approach is to maintain a cumulative matrix of information associating computer-internal tracks with physical objects. Updating this matrix is easy; furthermore, efficient methods were proposed for extracting concrete data association hypotheses from this representation. Empirical work using physical networks of camera arrays illustrated that our approach outperforms alternative paradigms that are commonly used throughout all of science.

Despite these advances, the work possesses a number of limitations. Specifically, our closed world assumption is problematic, although we believe the extension to open worlds is relatively straightforward. Also missing is a tight integration of our discrete formulation into continuous-valued traditional tracking algorithms such as EKFs. Such extensions warrant further research.

We believe the key innovation here is best understood from a graphical model perspective. Sampling $K$ good data associations *cannot* exploit conditional independence in the data association posterior, hence will always require that $K$ is an exponential function of $N$. The information form and the equivalent graphical network in Fig. 1e exploits conditional independences. This subtle difference makes it possible to get away with $O(N^2)$ memory and $O(N)$ computation without a loss of accuracy when $N$ increases, as shown in Fig. 3a. The information form discussed here—and the associated graphical networks—promise to overcome a key brittleness associated with the current state-of-the-art in online data association.

### Acknowledgements

We gratefully thank Jaewon Shin and Leo Guibas for helpful discussions.

This research was sponsored by the Defense Advanced Research Projects Agency (DARPA) under the ACIP program and grant number NBCH104009.

# References

[1] Y. Bar-Shalom and X.-R. Li. *Estimation and Tracking: Principles, Techniques, and Software*. YBS, Danvers, MA, 1998.

[2] F. Dellaert, S.M. Seitz, C. Thorpe, and S. Thrun. EM, MCMC, and chain flipping for structure from motion with unknown correspondence. *Machine Learning*, 50(1-2):45–71, 2003.

[3] A. Doucet, J.F.G. de Freitas, and N.J. Gordon, editors. *Sequential Monte Carlo Methods in Practice*. Springer, 2001.

[4] M. A. Fischler and R. C. Bolles. Random sample consensus: A paradigm for model fitting with applications to image analysis and automated cartography. *Communications of the ACM*, 24:381–395, 1981.

[5] P. Maybeck. *Stochastic Models, Estimation, and Control, Volume 1*. Academic Press, 1979.

[6] H. Pasula, S. Russell, M. Ostland, and Y. Ritov. Tracking many objects with many sensors. IJCAI-99.

[7] J. Pearl. *Probabilistic reasoning in intelligent systems: networks of plausible inference*. Morgan Kaufmann, 1988.

[8] J. Shin, N. Lee, S. Thrun, and L. Guibas. Lazy inference on object identities in wireless sensor networks. IPSN-05.

[9] D.B. Reid. An algorithm for tracking multiple targets. *IEEE Transactions on Aerospace and Electronic Systems*, AC-24:843–854, 1979.

[10] S. Thrun, Y. Liu, D. Koller, A.Y. Ng, Z. Ghahramani, and H. Durrant-Whyte. Simultaneous localization and mapping with sparse extended information filters. IJRR, 23(7/8), 2004.

[11] D. Fox, J. Hightower, L. Lioa, D. Schulz, and G. Borriello. Bayesian Filtering for Location Estimation. IEEE Pervasive Computing, 2003.
